# Identifying graph-structured activation patterns in networks

**James Sharpnack**
Machine Learning Department, Statistics Department
Carnegie Mellon University
Pittsburgh, PA 15213
jsharpna@andrew.cmu.edu

**Aarti Singh**
Machine Learning Department
Carnegie Mellon University
Pittsburgh, PA 15213
aartisingh@cmu.edu

## Abstract

We consider the problem of identifying an activation pattern in a complex, large-scale network that is embedded in very noisy measurements. This problem is relevant to several applications, such as identifying traces of a biochemical spread by a sensor network, expression levels of genes, and anomalous activity or congestion in the Internet. Extracting such patterns is a challenging task specially if the network is large (pattern is very high-dimensional) and the noise is so excessive that it masks the activity at any single node. However, typically there are statistical dependencies in the network activation process that can be leveraged to fuse the measurements of multiple nodes and enable reliable extraction of high-dimensional noisy patterns. In this paper, we analyze an estimator based on the graph Laplacian eigenbasis, and establish the limits of mean square error recovery of noisy patterns arising from a probabilistic (Gaussian or Ising) model based on an arbitrary graph structure. We consider both deterministic and probabilistic network evolution models, and our results indicate that by leveraging the network interaction structure, it is possible to consistently recover high-dimensional patterns even when the noise variance increases with network size.

## 1 Introduction

The problem of identifying high-dimensional activation patterns embedded in noise is important for applications such as contamination monitoring by a sensor network, determining the set of differentially expressed genes, and anomaly detection in networks. Formally, we consider the problem of identifying a pattern corrupted by noise that is observed at the $p$ nodes of a network:

$$y_i = x_i + \zeta_i \qquad i \in [p] = \{1, \dots, p\} \qquad (1)$$

Here $y_i$ denotes the observation at node $i$, $\mathbf{x} = [x_1, \dots, x_p] \in \mathbb{R}^p$ (or $\{0,1\}^p$) is the $p$-dimensional *unknown* continuous (or binary) activation pattern, and the noise $\zeta_i \overset{\text{iid}}{\sim} \mathcal{N}(0, \sigma^2)$, the Gaussian distribution with mean zero and variance $\sigma^2$. This problem is particularly challenging when the network is large-scale, and hence $\mathbf{x}$ is a high-dimensional pattern embedded in heavy noise. Classical approaches to this problem in the signal processing and statistics literature involve either thresholding the measurements at every node, or in the discrete case, matching the observed noisy measurements with all possible patterns (also known as the scan statistic). The first approach does not work well when the noise level is too high, rendering the per node activity statistically insignificant. In this case, multiple hypothesis testing effects imply that the noise variance needs to decrease as the number of nodes $p$ increase [10, 1] to enable consistent mean square error (MSE) recovery. The second approach based on the scan statistic is computationally infeasible in high-dimensional settings as the number of discrete patterns scale exponentially ($\geq 2^p$) in the number of dimensions $p$.

In practice, network activation patterns tend to be structured due to statistical dependencies in the network activation process. Thus, it is possible to recover activation patterns in a computationally and statistically efficient manner in noisy high-dimensional settings by leveraging the structure of

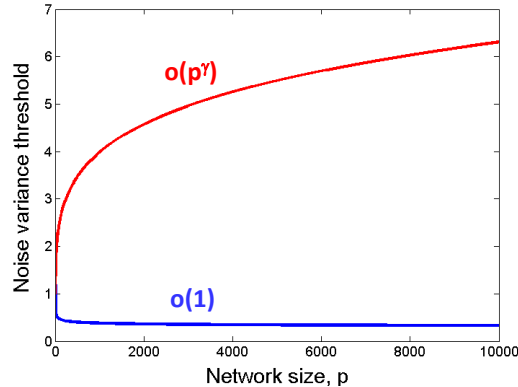

Figure 1: Threshold of noise variance below which consistent MSE recovery of network activation patterns is possible. If the activation is independent at each node, noise variance needs to decrease as network size $p$ increases (in blue). If dependencies in the activation process are harnessed, noise variance can increase as $p^\gamma$ where $0 < \gamma < 1$ depends on network interactions (in red).

the dependencies between node measurements. In this paper, we study the limits of MSE recovery of high-dimensional, graph-structured noisy patterns. Specifically, we assume that the patterns $\mathbf{x}$ are generated from a probabilistic model, either Gaussian graphical model (GGM) or Ising (binary), based on a general graph structure $G(V, E)$, where $V$ denotes the $p$ vertices and $E$ denotes the edges.

Gaussian graphical model:    $p(\mathbf{x}) \propto \exp(-\mathbf{x}^T \Sigma^{-1} \mathbf{x})$

Ising model:    $p(\mathbf{x}) \propto \exp\left(-\sum_{(i,j)\in E} W_{ij}(x_i - x_j)^2\right) \propto \exp(-\mathbf{x}^T \mathbf{L} \mathbf{x})$    (2)

In the Ising model, $\mathbf{L} = \mathbf{D} - \mathbf{W}$ denotes the graph Laplacian, where $\mathbf{W}$ is the weighted adjacency matrix and $\mathbf{D}$ is the diagonal matrix of node degrees $d_i = \sum_{j:(i,j)\in E} W_{ij}$. In the Gaussian graphical model, $\mathbf{L} = \Sigma^{-1}$ denotes the inverse covariance matrix whose zero entries indicate the absence of an edge between the corresponding nodes in the graph. The graphical model implies that all patterns are not equally likely to occur in the network. Patterns in which the values of nodes that are connected by an edge agree are more likely, the likelihood being determined by the weights $W_{ij}$ of the edges. Thus, the graph structure dictates the statistical dependencies in network measurements. We assume that this graph structure is known, either because it corresponds to the physical topology of the network or it can be learnt using network measurements [18, 25].

In this paper, we are concerned with the following problem: *What is the largest amount of noise that can be tolerated, as a function of the graph and parameters of the model, while allowing for consistent reconstruction of graph-structured network activation patterns?* If the activations at network nodes are independent of each other, the noise variance ($\sigma^2$) must decrease with network size $p$ to ensure consistent MSE recovery [10, 1]. We show that by exploiting the network dependencies, it is possible to consistently recover high-dimensional patterns when the noise variance is much larger (can grow with the network size $p$). See Figure 1.

We characterize the learnability of graph structured patterns based on the eigenspectrum of the network. To this end, we propose using an estimator based on thresholding the projection of the network measurements onto the graph Laplacian eigenvectors. This is motivated by the fact that in the Ising model, unlike the GGM, the Bayes rule and it's risk have no known closed form. Our results indicate that the noise threshold is determined by the eigenspectrum of the Laplacian. For the GGM this procedure reduces to PCA and the noise threshold depends on the eigenvalues of the covariance matrix, as expected. We show that for simple graph structures, such as hierarchical or lattice graphs, as well as the random Erdös-Rényi graph, the noise threshold can possibly grow in the network size $p$. Thus, leveraging the structure of network interactions can enable extraction of high-dimensional patterns embedded in heavy noise.

This paper is organized as follows. We discuss related work in Section 2. Limits of MSE recovery for graph-structured patterns are investigated in Section 3 for the binary Ising model, and in Section 4 for the Gaussian graphical model. In Section 5, we analyze the noise threshold for some simple deterministic and random graph structures. Simulation results are presented in Section 6, and concluding discussion in Section 7. Proof sketches are included in the Appendix.

## 2 Related work

Given a prior, the Bayes optimal estimators are known to be the posterior mean under MSE, the Maximum A Posterior (MAP) rule under 0/1 loss, and the posterior centroid under Hamming loss [8]. However, these estimators and their corresponding risks (expected loss) have no closed form for the Ising graphical model and are intractable to analyze. The estimator we propose based on the graph Laplacian eigenbasis is both easy to compute and analyze. Eigenbasis of the graph Laplacian has been successfully used for problems, such as clustering [20, 24], dimensionality reduction [5], and semi-supervised learning [4, 3]. The work on graph and manifold regularization [4, 3, 23, 2] is closely related and assumes that the function of interest is smooth with respect to the graph, which is essentially equivalent to assuming a graphical model prior of the form in Eq. (2). However, the use of graph Laplacian is theoretically justified mainly in the embedded setting [6, 21], where the data points are sampled from or near a low-dimensional manifold, and the graph weights are the distances between two points as measured by some kernel. To the best of our knowledge, no previous work studies the noise threshold for consistent MSE recovery of arbitrary graph-structured patterns.

There have been several attempts at constructing multi-scale basis for graphs that can efficiently represent localized activation patterns, notably diffusion wavelets [9] and treelets [17], however their approximation capabilities are not well understood. More recently, [22] and [14] independently proposed unbalanced Haar wavelets and characterized their approximation properties for tree-structured binary patterns. We argue in Section 5.1 that the unbalanced Haar wavelets are a special instance of graph Laplacian eigenbasis when the underlying graph is hierarchical. On the other hand, a lattice graph structure yields activations that are globally supported and smooth, and in this case the Laplacian eigenbasis corresponds to the Fourier transform (see Section 5.2). Thus, the graph Laplacian eigenbasis provides an efficient representation for patterns whose structure is governed by the graph.

## 3 Denoising binary graph-structured patterns

The binary Ising model is essentially a discrete version of the GGM, however, the Bayes rule and risk for the Ising model have no known closed form. For binary graph-structured patterns drawn from an Ising prior, we suggest a different estimator based on projections onto the graph Laplacian eigenbasis. Let the graph Laplacian $\mathbf{L}$ have spectral decomposition, $\mathbf{L} = \mathbf{U}\Lambda\mathbf{U}^T$, and denote the first $k$ eigenvectors (corresponding to the smallest eigenvalues) of $\mathbf{L}$ by $\mathbf{U}_{[k]}$. Define the estimator

$$\widehat{\mathbf{x}}_k = \mathbf{U}_{[k]}\mathbf{U}_{[k]}^T\mathbf{y}, \tag{3}$$

which is a hard thresholding of the projection of network measurements $\mathbf{y} = [y_1, \ldots, y_p]$ onto the graph Laplacian eigenbasis. The following theorem bounds the MSE of this estimator.

**Theorem 1.** *The Bayes MSE of the estimator in Eq. (3) for the observation model in Eq. (1), when the binary activation patterns are drawn from the Ising prior of Eq. (2) is bounded as*

$$R_B := \frac{1}{p}\,\mathbb{E}[\|\widehat{\mathbf{x}}_k - \mathbf{x}\|^2] \le \min\left(1, \frac{\delta}{\lambda_{k+1}}\right) + \frac{k\sigma^2}{p} + e^{-p}$$

*where $0 < \delta < 2$ is a constant and $\lambda_{k+1}$ is the $(k+1)^{th}$ smallest eigenvalue of $\mathbf{L}$.*

Through this bias-variance decomposition, we see the eigenspectrum of the graph Laplacian determines a bound on the MSE for binary graph-structured activations. In practice, $k$ can be chosen using FDR[1] in the eigendomain or cross-validation.

**Remark:** Consider the binarized estimator $\widehat{\mathbf{x}}_i' = \mathbf{1}_{\widehat{\mathbf{x}}_i > 1/2}, i \in [p]$. Then the results of Theorem 1 also provide an upper bound on the expected Hamming distance of this new estimator since $\mathbb{E}[d_H(\widehat{\mathbf{x}}', \mathbf{x})] = \text{MSE}(\widehat{\mathbf{x}}') \le 4\text{MSE}(\widehat{\mathbf{x}})$, by the triangle inequality.

## 4 Denoising Gaussian graph-structured patterns

If the network activation patterns are generated by a Gaussian graphical model, it is easy to see that the eigenvalues of the Laplacian (inverse covariance) determine the MSE decay. Consider the GGM prior as in Eq. (2), then the posterior distribution is

$$\mathbf{x}|\mathbf{y} \sim \mathcal{N}\left((2\sigma^2\mathbf{L} + \mathbf{I})^{-1}\mathbf{y}, \left(2\mathbf{L} + \sigma^{-2}\mathbf{I}\right)^{-1}\right), \tag{4}$$

where $\mathbf{I}$ is the identity matrix. The posterior mean is the Bayes optimal estimator with Bayes MSE, $\frac{1}{p}\sum_{i\in[p]}(2\lambda_i + \sigma^{-2})^{-1}$, where $\{\lambda_i\}_{i\in[p]}$ are the ordered eigenvalues of $\mathbf{L}$. For the GGM, we obtain a result similar to Theorem 1 for the sake of bounding the performance of the Bayes rule.

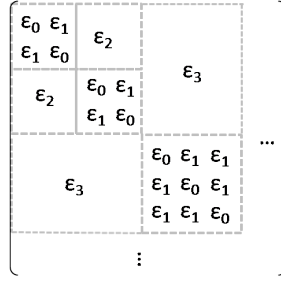

Figure 2: Weight matrices corresponding to hierarchical dependencies between node variables.

**Theorem 2.** *The Bayes MSE of the estimator in Eq. (3) for the observation model in Eq. (1), when the activation patterns are drawn from the Gaussian graphical model prior of Eq. (2) is bounded as*

$$R_B := \frac{1}{p}\mathbb{E}[\|\widehat{\mathbf{x}}_k - \mathbf{x}\|^2] = \frac{1}{p}\sum_{i=k+1}^{p}\frac{1}{2\lambda_i} + \frac{k\sigma^2}{p} \leq \frac{1}{2\lambda_{k+1}} + \frac{k\sigma^2}{p}$$

Hence, the Bayes MSE for the estimator of Eq. (3) under the GGM or Ising prior is bounded above by $2/\lambda_k + \sigma^2 k/p + e^{-p}$ which is the form used to prove Corollaries 1, 2, 3 in the next section.

## 5 Noise threshold for some simple graphs

In this section, we discuss the eigenspectrum of some simple graphs and use the MSE bounds derived in the previous section to analyze the amount of noise that can be tolerated while ensuring consistent MSE recovery of high-dimensional patterns. In all these examples, we find that the tolerable noise level scales as $\sigma^2 = o(p^\gamma)$, where $\gamma \in (0, 1)$ characterizes the strength of network interactions.

### 5.1 Hierarchical structure

Consider that, under an appropriate permutation of rows and columns, the weight matrix $\mathbf{W}$ has the hierarchical block form shown in Figure 2. This corresponds to hierarchical graph structured dependencies between node variables, where $\epsilon_\ell > \epsilon_{\ell+1}$ denote the strength of interactions between nodes that are in the same block at level $\ell = 0, 1, \ldots, L$. It is easy to see that in this case the eigenvectors $\mathbf{u}$ of the graph Laplacian correspond to unbalanced Haar wavelet basis (proposed in [22, 14]), i.e. $\mathbf{u} \propto \frac{1}{|c_2|}\mathbf{1}_{c_2} - \frac{1}{|c_1|}\mathbf{1}_{c_1}$, where $c_1$ and $c_2$ are groups of variables within blocks at the same level that are merged together at the next level (see [19] for the case of a full dyadic hierarchy).

**Lemma 1.** *For a dyadic hierarchy with $L$ levels, the eigenvectors of the graph Laplacian are the standard Haar wavelet basis and there are $L + 1$ unique eigenvalues with the smallest eigenvalue $\lambda_0 = 0$, and the $\ell^{th}$ smallest unique eigenvalue ($\ell \in [L]$) is $2^{\ell-1}$-fold degenerate and given as*

$$\lambda_\ell = \sum_{i=L-\ell+1}^{L} 2^{i-1}\epsilon_i + 2^{L-\ell}\epsilon_{L-\ell+1}.$$

Using the bound on MSE as given in Theorems 1 and 2, we can now derive the noise threshold that allows for consistent MSE recovery of high-dimensional patterns as the network size $p \to \infty$.

**Corollary 1.** *Consider a graph-structured pattern drawn from an Ising model or the GGM with weight matrix $\mathbf{W}$ of the hierarchical block form as depicted in Figure 2. If $\epsilon_\ell = 2^{-\ell(1-\beta)} \, \forall \ell \leq \gamma \log_2 p + 1$, for constants $\gamma, \beta \in (0, 1)$, and $\epsilon_\ell = 0$ otherwise, then the noise threshold for consistent MSE recovery ($R_B = o(1)$) is*

$$\sigma^2 = o(p^\gamma).$$

Thus, if we take advantage of the network interaction structure, it is possible to tolerate noise with variance that scales with the network size $p$, whereas without exploiting structure the noise variance needs to decrease with $p$, as discussed in the introduction. Larger $\gamma$ implies stronger network interactions, and hence larger the noise threshold.

### 5.2 Regular Lattice structure

Now consider the lattice graph which is constructed by placing vertices in a regular grid on a $d$ dimensional torus and adding edges of weight 1 to adjacent points. Let $p = n^d$. For $d = 1$

this is a cycle which has a circulant weight matrix $w$, with eigenvalues $\{2\cos(\frac{2\pi k}{p}) : k \in [p]\}$ and eigenvectors correspond to the discrete Fourier transform [13]. Let $i = (i_1, ..., i_d), j = (j_1, ..., j_d) \in [n]^d$. Then the weight matrix of the lattice in $d$ dimensions is

$$W_{i,j} = w_{i_1,j_1}\delta_{i_2,j_2}...\delta_{i_d,j_d} + ... + w_{i_d,j_d}\delta_{i_1,j_1}...\delta_{i_{d-1},j_{d-1}} \tag{5}$$

where $\delta$ is the Kronecker delta function. This form for $\mathbf{W}$ and since all nodes have same degree gives us a closed form for the eigenvalues of the Laplacian, along with a concentration inequality.

**Lemma 2.** *Let $\lambda_\bullet^\mathbf{L}$ be an eigenvalue of the Laplacian, $\mathbf{L}$, of the lattice graph in $d$ dimensions with $p = n^d$ vertices, chosen uniformly at random. Then*

$$\mathbb{P}\{\lambda_\bullet^\mathbf{L} \leq d\} \leq \exp\{-d/8\}. \tag{6}$$

Hence, we can choose k such that $\lambda_k^\mathbf{L} \geq d$ and $k = \lceil pe^{-d/8} \rceil$. So, the risk bound becomes $\mathcal{O}(2/d + \sigma^2 e^{-d/8} + e^{-p})$, and as we increase dimensions of the lattice the MSE decays linearly.

**Corollary 2.** *Consider a graph-structured pattern drawn from an Ising model or GGM based on a lattice graph in $d$ dimensions with $p = n^d$ vertices. If $n$ is a constant and $d = 8\gamma \ln p$, for some constant $\gamma \in (0, 1)$, then the noise threshold for consistent MSE recovery ($R_B = o(1)$) is given as:*

$$\sigma^2 = o(p^\gamma).$$

Again, the noise variance can increase with the network size $p$, and larger $\gamma$ implies stronger network interactions as each variables interacts with more number of neighbors ($d$ is larger).

### 5.3 Erdös-Rényi random graph structure

Erdös-Rényi (ER) random graphs are generated by adding edges with weight $1$ between any two vertices within the vertex set $V$ (of size $p$) with probability $q_p$. It is known that the probability of edge inclusion ($q_p$) determines large geometric properties of the graph [11]. Real world networks are generally sparse, so we set $q_p = p^{-(1-\gamma)}$, where $\gamma \in (0, 1)$. Larger $\gamma$ implies higher probability of edge inclusion and stronger network interaction structure. Using the degree distribution [7], and a result from perturbation theory, we bound the quantiles of the eigenspectrum of $\mathbf{L}$.

**Lemma 3.** *Let $\lambda_\bullet$ denote an eigenvalue of $\mathbf{L}$ chosen uniformly at random. Let $\mathbb{P}_G$ be the probability measure induced by the ER random graph and $\mathbb{P}_\bullet$ be the uniform distribution over eigenvalues conditional on the graph. Then, for any $\alpha_p$ increasing in $p$,*

$$\mathbb{P}_G\{\mathbb{P}_\bullet\{\lambda_\bullet \leq p^\gamma/2 - p^{\gamma-1}\} \geq \alpha_p p^{-\gamma}\} = \mathcal{O}(1/\alpha_p) \tag{7}$$

Hence, we are able to set the sequence of quantiles for the eigenvalue distribution $k_p = \lceil \alpha_p p^{1-\gamma} \rceil$ such that $\mathbb{P}_G\{\lambda_{k_p} \leq p^\gamma/2 - p^{\gamma-1}\} = \mathcal{O}(1/\alpha_p)$. So, we obtain a bound for the expected Bayes MSE (with respect to the graph) $\mathbb{E}_G[R_B] \leq \mathcal{O}(p^{-\gamma}) + \sigma^2 \mathcal{O}(\alpha_p p^{-\gamma}) + \mathcal{O}(1/\alpha_p)$.

**Corollary 3.** *Consider a graph $G$ drawn from an Erdös-Rényi random graph model with $p$ vertices and probability of edge inclusion $q_p = p^{-(1-\gamma)}$ for some constant $\gamma \in (0, 1)$. If the latent graph-structured pattern is drawn from an Ising model or a GGM with the Laplacian of $G$, then the noise variance that can be tolerated while ensuring consistent MSE recovery ($R_B = o_{\mathbb{P}_G}(1)$) is given as:*

$$\sigma^2 = o(p^\gamma).$$

## 6 Experiments

We simulate patterns from the Ising model defined on hierarchical, lattice and ER graphs. Since the Ising distribution admits a closed form for the distribution of one node conditional on the rest of the nodes, a Gibbs sampler can be employed. Histograms of the eigenspectrum for the hierarchical tree graph with a large depth, the lattice graph in high dimensions, and a draw from the ER graph with many nodes is shown in figures 3(a), 4(a), 5(a) respectively. The eigenspectrum of the lattice and ER graphs illustrate the concentration of the eigenvalues about the expected degree of each node.

We use iterative eigenvalue solvers to form our estimator and choose the quantile $k$ by minimizing the bound in Theorem 1. We compute the Bayes MSE (by taking multiple draws) of our estimator for a noisy sample of node measurements. We observe in all of the models that the eigenmap estimator is a substantial improvement over Naive (the Bayes estimator that ignores the structure).

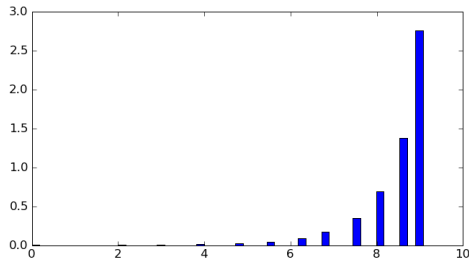

(a) Eigenvalue Histogram for hierarchical tree.

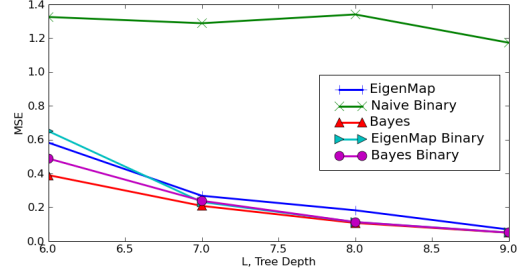

(b) Estimator Performance

Figure 3: The eigenvalue histogram for the binary tree, $L = 11$, $\beta = .1$ (left) and the performance of various estimators (right) with $\beta = 0.05$ and $\sigma^2 = 4$, both with $\gamma = 1$.

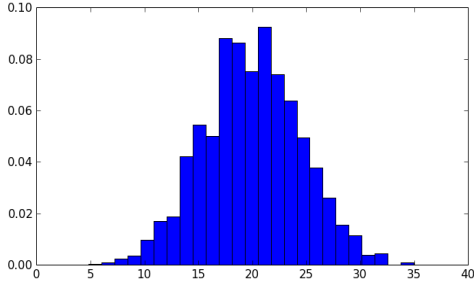

(a) Eigenvalue Histogram for Lattice.

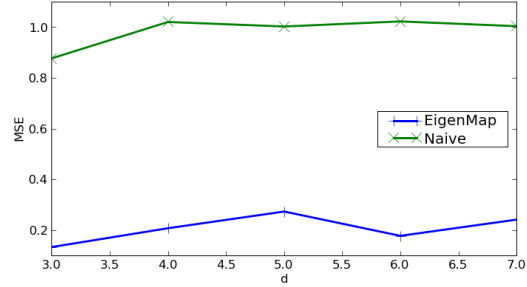

(b) Estimator Performance

Figure 4: The eigenvalue histogram for the lattice with $d = 10$ and $p = 5^{10}$ (left) and estimator performances (right) with $p = 3^d$ and $\sigma^2 = 1$. Notice that the eigenvalues concentrate around $2d$.

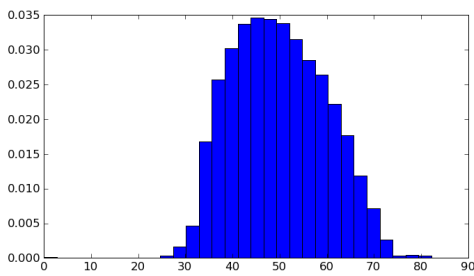

(a) Eigenvalue Histogram for Erdös-Rényi.

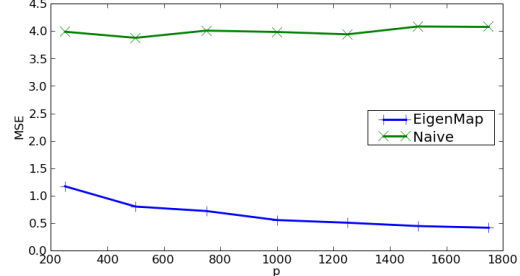

(b) Estimator Performance

Figure 5: The eigenvalue histogram for a draw from the ER graph with $p = 2500$ and $q_p = p^{-.5}$ (left) and the estimator performances (right) with $q_p = p^{-.75}$ and $\sigma^2 = 4$. Notice that the eigenvalues are concentrated around $p^\gamma$ where $q_p = p^{-(1-\gamma)}$.

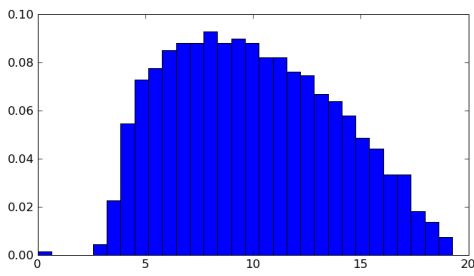

(a) Eigenvalue Histogram for Watts-Strogatz.

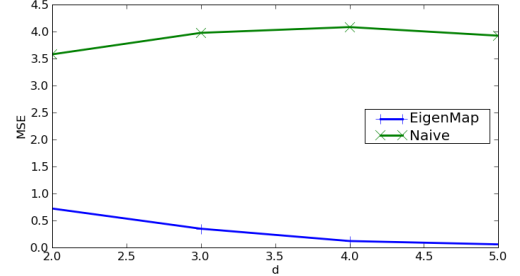

(b) Estimator Performance

Figure 6: The eigenvalue histogram for a draw from the Watts-Strogatz graph with $d = 5$ and $p = 4^5$ with $0.25$ probability of rewiring (left) and estimator performances (right) with $4^d$ vertices and $\sigma^2 = 4$. Notice that the eigenvalues are concentrated around $2d$.

See Figures 3(b), 4(b), 5(b). For the hierarchical model, we also sample from the posterior using a Gibbs sampler and estimate the posterior mean (Bayes rule under MSE). We find that the posterior mean is only a slight improvement over the eigenmap estimator (Figure 3(b)), despite it's difficulty to compute. Also, a binarized version of these estimators does not substantially change the MSE.

We also simulate graphs from the Watts-Strogatz 'small world' model [26], which is known to be an appropriate model for self-organizing systems such as biological systems and human networks. The 'small world' graph is generated by forming the lattice graph described in Section 5.2, then rewiring each edge with some constant probability to another vertex uniformly at random such that loops are never created. We observe that the eigenvalues concentrate (more tightly than the lattice graph) around the expected degree $2d$ (Figure 6(a)) and note that, like the ER model, the eigenspectrum converges to a nearly semi-circular distribution [12]. Similarly, the MSE decays in a fashion similar to the ER model (Figure 6(b)).

# 7  Discussion

In this paper, we have characterized the improvement in noise threshold, below which consistent MSE recovery of high-dimensional network activation patterns embedded in heavy noise is possible, as a function of the network size and parameters governing the statistical dependencies in the activation process. Our results indicate that by leveraging the network interaction structure, it is possible to tolerate noise with variance that increases with the size of the network whereas without exploiting dependencies in the node measurements, the noise variance needs to decrease as the network size grows to accommodate for multiple hypothesis testing effects.

While we have only considered MSE recovery, it is often possible to detect the presence of patterns in much heavier noise, even though the activation values may not be accurately recovered [16]. Establishing the noise threshold for detection, deriving upper bounds on the noise threshold, and extensions to graphical models with higher-order interaction terms are some of the directions for future work. In addition, the thresholding estimator based on the graph Laplacian eigenbasis can also be used in high-dimensional linear regression or compressed sensing framework to incorporate structure, in addition to sparsity, of the relevant variables.

# Appendix

**Proof sketch of Theorem 1:** First, we argue that whp, $\mathbf{x}^T\mathbf{L}\mathbf{x} \leq \delta p$, where $0 < \delta < 2$ is a constant. Let $\Omega = \{\mathbf{x} : \mathbf{x}^T\mathbf{L}\mathbf{x} \leq \delta p\}$ and $\bar{\Omega}$ denotes its complement. By Markov's inequality, for $t > 0$,

$$\mathbb{P}\{\mathbf{x}^T\mathbf{L}\mathbf{x} > C\} = \mathbb{P}\{e^{t\mathbf{x}^T\mathbf{L}\mathbf{x}} > e^{tC}\} \leq e^{-tC}\,\mathbb{E}e^{t\mathbf{x}^T\mathbf{L}\mathbf{x}}$$

Let $\nu$ denote the uniform distribution over $\{0,1\}^p$ and $N(\mathbf{L}) = \int \nu(dx)e^{-\mathbf{x}^T\mathbf{L}\mathbf{x}}$. Then,

$$\mathbb{E}e^{\mathbf{x}^T(t\mathbf{L})\mathbf{x}} = \int \nu(dx)N(\mathbf{L})^{-1}e^{-\mathbf{x}^T\mathbf{L}\mathbf{x}}e^{\mathbf{x}^T(t\mathbf{L})\mathbf{x}} = \frac{\int \nu(dx)e^{-\mathbf{x}^T\mathbf{L}(1-t)\mathbf{x}}}{N(\mathbf{L})} = \frac{N((1-t)\mathbf{L})}{N(\mathbf{L})} \leq 2^p$$

where the last step follows since $N(\mathbf{L}) = \sum_{x \in \{0,1\}^p} e^{-x^T\mathbf{L}x}$ and $\mathbf{L}\vec{1} = 0$ implying that $1 \leq N(\mathbf{L}), N((1-t)\mathbf{L}) \leq 2^p, \forall t \in (0,1)$. This gives us the Chernoff-type bound,

$$\mathbb{P}(\bar{\Omega}) \leq \mathbb{P}\{\mathbf{x}^T\mathbf{L}\mathbf{x} > C\} \leq e^{-tC}2^p = e^{(\log 2 - tC/p)p} \leq e^{-p}$$

by setting $C = \delta p$ and $\delta = \frac{1+\log 2}{t}$. If we choose $t < \frac{1+\log 2}{2}$ then $\delta < 2$.

Let $\mathbf{u}_i$ denote the $i^{th}$ eigenvector of the graph Laplacian $\mathbf{L}$, then under this orthonormal basis,

$$\mathbb{E}[\|\widehat{\mathbf{x}}_k - \mathbf{x}\|^2] \leq \mathbb{E}[\sum_{i=k+1}^{p} \mathbf{u}_i^T\mathbf{x}^2 \,|\, \Omega] + pP(\bar{\Omega}) + k\sigma^2 \leq \sup_{\mathbf{x}:\mathbf{x}^T\mathbf{L}\mathbf{x} \leq \delta p} \sum_{i=k+1}^{p} \mathbf{u}_i^T\mathbf{x}^2 + p\,e^{-p} + k\sigma^2.$$

We now establish that $\sup_{\mathbf{x}:\mathbf{x}^T\mathbf{L}\mathbf{x} \leq \delta p} \sum_{i=k+1}^{p}(\mathbf{u}_i^T\mathbf{x})^2 \leq p\min(1, \delta/\lambda_{k+1})$, and the result follows. Let $\tilde{\mathbf{x}}_i = \mathbf{u}_{i+k}^T\mathbf{x}, i \in [p-k]$ and note that $\mathbf{x}^T\mathbf{L}\mathbf{x} = \sum_{i=1}^{p} \lambda_i(\mathbf{u}_i^T\mathbf{x})^2 \geq \sum_{i=k+1}^{p} \lambda_i\tilde{\mathbf{x}}_i^2$, for $\lambda_i$ the $i^{th}$ eigenvalue of $\mathbf{L}$. Consider the primal problem,

$$\max \sum_{j=1}^{p-k} \tilde{\mathbf{x}}_j^2 \text{ such that } \sum_{j=1}^{p-k} \lambda_j\tilde{\mathbf{x}}_j^2 \leq \delta p, \ \tilde{\mathbf{x}} \in \mathbb{R}^{p-k}$$

Note that $\mathbf{x}$ contained within the ellipsoid $\mathbf{x}^T\mathbf{L}\mathbf{x} \leq \delta p$, $\mathbf{x} \in \{0,1\}^p$ implies that $\tilde{\mathbf{x}}$ is feasible, so a solution to the optimization upper bounds $\sup_{\mathbf{x}:\mathbf{x}^T\mathbf{L}\mathbf{x}\leq\delta p} \sum_{i=k+1}^p (\mathbf{u}_i^T\mathbf{x})^2$. By forming the dual problem, we find that the solution, $\mathbf{x}^*$, to the primal problem attains a bound of $||\tilde{\mathbf{x}}||^2 \leq ||\tilde{\mathbf{x}}^*||^2 = \delta p/\lambda_{k+1}$. Also, $||\tilde{\mathbf{x}}||^2 \leq ||\mathbf{x}||^2 \leq p$, so we obtain the desired bound.

**Proof sketch of Theorem 2:** Under the same notation as the previous proof, notice that $\mathbf{u}_i^T\mathbf{x} \sim N(0, (2\lambda)_i^{-1})$ independently over $i \in [p]$. Then $\mathbb{E}||\tilde{\mathbf{x}}||^2 = \sum_{i=k+1}^p (2\lambda_i)^{-1}$ and, so, $\frac{1}{p}\mathbb{E}||\hat{\mathbf{x}} - \mathbf{x}||^2 = \frac{1}{p}\mathbb{E}||\tilde{\mathbf{x}}||^2 + \frac{1}{p}\mathbb{E}||\mathbf{U}_{[k]}\zeta||^2 = \frac{1}{p}\sum_{i=k+1}^p (2\lambda_i)^{-1} + \sigma^2 k/p \leq (2\lambda_{k+1})^{-1} + \sigma^2 k/p$.

**Proof sketch of Corollary 1:** Let $\ell^* = (1-\gamma)\log_2 p$. Since $\epsilon_i = 2^{-i(1-\beta)}$ $\forall i < L - \ell^* + 1$ and $\epsilon_i = 0$ otherwise, we have for $\ell \geq \ell^*$ and since $L = \log_2 p$, $\lambda_\ell \geq 2^{\beta(L-\ell^*)}2^{\beta-1} = p^{\beta\gamma}2^{\beta-1}$, which is increasing in $p$. Therefore, we can pick $k = 2^{\ell^*}$ and since $2^{\ell^*}/p = p^{-\gamma}$, the result follows.

**Proof sketch of Lemma 2:** If $v_1, ..., v_d$ are a subset of the eigenvectors of $w$ with eigenvalues $\lambda_1, ..., \lambda_d$, then $W(v_1 \otimes ... \otimes v_d) = (\lambda_1 + ... + \lambda_d)(v_1 \otimes ... \otimes v_d)$ where $\otimes$ denotes tensor product. Noting that the $D_{ii} = 2d, \forall i \in [n]^d$ then we see that the Laplacian $\mathbf{L}$ has eigenvalues $\lambda_i^L = 2d - \lambda_i^W = \sum_j^{[d]}(2 - \lambda_{i_j}^w)$ for all $i \in [n]^d$. Recall $\lambda_k^w = 2\cos(\frac{2\pi k}{n})$ for some $k \in [n]$. Let $i$ be distributed uniformly over $[n]^d$. Then $\mathbb{E}[\lambda_{i_j}^w] = 0$, and by Hoeffding's inequality,

$$\mathbb{P}\{\sum_{j=1}^d (2 - \lambda_{i_j}^w) - 2d \leq -t\} \leq \exp\{-2t^2/16d\}$$

So, using $t = d$ we get that $\mathbb{P}\{\sum_{j=1}^d (2 - \lambda_{i_j}^w) \leq d\} \leq \exp\{\frac{-d}{8}\}$ and the result follows.

**Proof of Lemma 3:** We introduce a random variable $\bullet$ that is uniform over $[p]$. Note that, conditioned on this random variable, $d_\bullet \sim \text{Binomial}(p - 1, q_p)$ and $\text{Var}(d_\bullet) \leq pq_p$. We decompose the Laplacian, $\mathbf{L} = \mathbf{D} - \mathbf{W} = (\bar{d}\mathbf{I} - \mathbf{W}) + (\mathbf{D} - \bar{d}\mathbf{I})$, into the expected degree of each vertex ($\bar{d} = (p-1)q_p$), $\mathbf{W}$ and the deviations from the expected degree and use the following lemma.

**Lemma 4** (Wielandt-Hoffman Theorem). *[15, 27] Suppose $A = B + C$ are symmetric $p \times p$ matrices and denote the ordered eigenvalues by $\{\lambda_i^A, \lambda_i^B\}_{i=1}^p$. If $||.||_F$ denotes the Frobenius norm,*

$$\sum_{i=1}^p (\lambda_i^A - \lambda_i^B)^2 \leq ||C||_F^2 \qquad (8)$$

Notice that $\mathbb{E}_G||\mathbf{D} - \bar{d}\mathbf{I}||_F^2/p = \text{Var}(d_\bullet)$ and so $\mathbb{E}_G||\lambda^{\bar{d}\mathbf{I}-\mathbf{W}} - \lambda^L||^2/p \leq pq_p = p^\gamma$ *(i)*. Also, it is known that for $\gamma \in (0, 1)$ the eigenvalues converge to a semicircular distribution[12] such that $\mathbb{P}_G\{|\lambda_\bullet^W| \leq 2\sqrt{pq_p(1 - q_p)}\} \to 1$. Since $2\sqrt{pq_p(1 - q_p)} \leq 2p^{\gamma/2}$, we have $\mathbb{E}_G[(\lambda_\bullet^W)^2] \leq 4p^\gamma$ for large enough $p$ *(ii)*. Using triangle inequality,

$$\mathbb{E}_G[(\lambda_\bullet^L - (p-1)q_p)^2] \leq \mathbb{E}_G[(\lambda_\bullet^L - ((p-1)q_p - \lambda_\bullet^W))^2] + \mathbb{E}_G[(\lambda_\bullet^W)^2] \leq 5p^\gamma, \qquad (9)$$

where the last step follows using *(i)*, *(ii)* and $\lambda_i^{\bar{d}\mathbf{I}-\mathbf{W}} = (p-1)q_p - \lambda_i^W$. By Markov's inequality,

$$\mathbb{P}_G\{\mathbb{P}_\bullet\{\lambda_\bullet^L \leq \frac{p^\gamma}{2} - p^{\gamma-1}\} \geq \alpha_p p^{-\gamma}\} \leq \frac{p^\gamma}{\alpha_p}\mathbb{E}_G[\mathbb{P}_\bullet\{\lambda_\bullet^L \leq \frac{p^\gamma}{2} - p^{\gamma-1}\}] \qquad (10)$$

for any $\alpha_p$ which is an increasing positive function in $p$. We now analyze the right hand side.

$$\mathbb{P}_\bullet\{|\lambda_\bullet^L - (p-1)q_p| \geq \epsilon\} \leq \epsilon^{-2}\mathbb{E}_\bullet[(\lambda_\bullet^L - (p-1)q_p)^2]$$

Note that $\mathbb{P}_\bullet\{\lambda_\bullet^L \leq pq_p - q_p - \epsilon\} \leq \mathbb{P}_\bullet\{|\lambda_\bullet^L - (p-1)q_p| \geq \epsilon\}$ and setting $\epsilon = pq_p/2 = p^\gamma/2$,

$$\mathbb{P}_\bullet\{\lambda_\bullet^L \leq p^\gamma/2 - p^{\gamma-1}\} \leq 4p^{-2\gamma}\mathbb{E}_\bullet[(\lambda_\bullet^L - (p-1)q_p)^2].$$

Hence, we are able to complete the lemma, such that for $p$ large enough, using Eqs. (10) and (9)

$$\mathbb{P}_G\{\mathbb{P}_\bullet\{\lambda_\bullet^L \leq \frac{p^\gamma}{2} - p^{\gamma-1}\} \geq \alpha_p p^{-\gamma}\} \leq \frac{4}{\alpha_p p^\gamma}\mathbb{E}_G[\mathbb{E}_\bullet[(\lambda_\bullet^L - (p-1)q_p)^2]] \leq \frac{20}{\alpha_p}. \qquad (11)$$

**Proof sketch of Corollary 3:** By lemma 3 and appropriately specifying the quantiles,

$$\mathbb{E}_G R_B \leq \mathbb{E}_G\left[\frac{2}{\lambda_{k_p}} + \sigma^2\frac{k_p}{p} + e^{-p}\right] \leq \left(\frac{2}{p^\gamma/2 - p^{\gamma-1}} + \sigma^2\mathcal{O}(\alpha_p p^{-\gamma}) + e^{-p}\right) + \mathcal{O}(\frac{1}{\alpha_p}) \quad (12)$$

Note that we have the freedom to choose $\alpha_p = \sqrt{p^\gamma/\sigma^2}$ making $\sigma^2\mathcal{O}(\alpha_p p^{-\gamma}) = \mathcal{O}(\sqrt{\sigma^2/p^\gamma}) = o(1)$ and $\mathcal{O}(1/\alpha_p) = o(1)$ if $\sigma^2 = o(p^\gamma)$.

# References

[1] F. Abramovich, Y. Benjamini, D. L. Donoho, and I. M. Johnstone, *Adapting to unknown sparsity by controlling the false discovery rate*, Annals of Statistics **34** (2006), no. 2, 584–653.

[2] Rie K. Ando and Tong Zhang, *Learning on graph with laplacian regularization*, Advances in Neural Information Processing Systems (NIPS), 2006.

[3] M. Belkin and P. Niyogi, *Semi-supervised learning on riemannian manifolds*, Machine Learning **56(1-3)** (2004), 209–239.

[4] Mikhail Belkin, Irina Matveeva, and Partha Niyogi, *Regularization and semi-supervised learning on large graphs*, Conference on Learning Theory (COLT), 2004.

[5] Mikhail Belkin and Partha Niyogi, *Laplacian eigenmaps for dimensionality reduction and data representation*, Neural Computation **15** (2003), no. 6, 1373–1396.

[6] ———, *Convergence of laplacian eigenmaps*, Advances in Neural Information Processing Systems (NIPS), 2006.

[7] B. Bollobas, *Random graphs*, Cambridge University Press, 2001.

[8] Luis E. Carvalho and Charles E. Lawrence, *Centroid estimation in discrete high-dimensional spaces with applications in biology*, PNAS **105** (2008), no. 9, 3209–3214.

[9] R. Coifman and M. Maggioni, *Diffusion wavelets*, Applied and Computational Harmonic Analysis **21** (2006), no. 1, 53–94.

[10] D. L. Donoho, I. M. Johnstone, J. C. Hoch, and A. S. Stern, *Maximum entropy and the nearly black object*, Journal of Royal Statistical Society, Series B **54** (1992), 41–81.

[11] P. Erdös and A Rényi, *On the evolution of random graphs*, Publication of the Mathematical Institute of the Hungarian Academy of Sciences, 1960, pp. 17–61.

[12] Illés J. Farkas, Imre Derényi, Albert-László Barabási, and Tamás Vicsek, *Spectra of real-world graphs: Beyond the semi-circle law*, Physical Review E **64** (2001), 1–12.

[13] Bernard Friedman, *Eigenvalues of composite matrices*, Mathematical Proceedings of the Cambridge Philosophical Society **57** (1961), 37–49.

[14] M. Gavish, B. Nadler, and R. Coifman, *Multiscale wavelets on trees, graphs and high dimensional data: Theory and applications to semi supervised learning*, 27th International Conference on Machine Learning (ICML), 2010.

[15] S. Jalan and J. N. Bandyopadhyay, *Random matrix analysis of network laplacians*, Tech. Report cond-mat/0611735, Nov 2006.

[16] J. Jin and D. L. Donoho, *Higher criticism for detecting sparse heterogeneous mixtures*, Annals of Statistics **32** (2004), no. 3, 962–994.

[17] A. B. Lee, B. Nadler, and L. Wasserman, *Treelets - an adaptive multi-scale basis for sparse unordered data*, Annals of Applied Statistics **2** (2008), no. 2, 435–471.

[18] N. Meinshausen and P. Buhlmann, *High dimensional graphs and variable selection with the lasso*, Annals of Statistics **34** (2006), no. 3, 1436–1462.

[19] A. T. Ogielski and D. L. Stein, *Dynamics on ultrametric spaces*, Physical Review Letters **55** (1985), 1634–1637.

[20] J. Shi and J. Malik, *Normalized cuts and image segmentation*, IEEE Trans. Pattern Analysis and Machine Intelligence **22** (2000), 888–905.

[21] A. Singer, *From graph to manifold laplacian: the convergence rate*, Applied and Computational Harmonic Analysis **21** (2006), no. 1, 135–144.

[22] A. Singh, R. Nowak, and R. Calderbank, *Detecting weak but hierarchically-structured patterns in networks*, 13th International Conference on Artificial Intelligence and Statistics (AISTATS), 2010.

[23] A. Smola and R. Kondor, *Kernels and regularization on graphs*, Conference on Learning Theory (COLT), 2003.

[24] Ulrike von Luxburg, *A tutorial on spectral clustering*, Statistics and Computing **17** (2007), no. 4, 395–416.

[25] M. Wainwright, P. Ravikumar, and J. D. Lafferty, *High-dimensional graphical model selection using $\ell_1$-regularized logistic regression*, Advances in Neural Information Processing Systems (NIPS), 2006.

[26] Duncan J. Watts and Steven H. Strogatz, *Collective dynamics of 'small-world' networks*, Nature **393** (1998), no. 6684, 440–442.

[27] Choujun Zhan, Guanrong Chen, and Lam F. Yeung, *On the distribution of laplacian eigenvalues versus node degrees in complex networks*, Physica A **389** (2010), 1779–1788.

